# MIME: Mutual Information Minimization and Entropy Maximization for Bayesian Belief Propagation

**Anand Rangarajan**

Dept. of Computer and Information Science and Engineering
University of Florida
Gainesville, FL 32611-6120, US
*anand@cise.ufl.edu*

**Alan L. Yuille**

Smith-Kettlewell Eye Research Institute
2318 Fillmore St.
San Francisco, CA 94115, US
*yuille@ski.org*

## Abstract

Bayesian belief propagation in graphical models has been recently shown to have very close ties to inference methods based in statistical physics. After Yedidia *et al.* demonstrated that belief propagation fixed points correspond to extrema of the so-called Bethe free energy, Yuille derived a double loop algorithm that is guaranteed to converge to a local minimum of the Bethe free energy. Yuille's algorithm is based on a certain decomposition of the Bethe free energy and he mentions that other decompositions are possible and may even be fruitful. In the present work, we begin with the Bethe free energy and show that it has a principled interpretation as pairwise mutual information minimization and marginal entropy maximization (MIME). Next, we construct a family of free energy functions from a spectrum of decompositions of the original Bethe free energy. For each free energy in this family, we develop a new algorithm that is guaranteed to converge to a local minimum. Preliminary computer simulations are in agreement with this theoretical development.

## 1 Introduction

In graphical models, Bayesian belief propagation (BBP) algorithms often (but not *always*) yield reasonable estimates of the marginal probabilities at each node [6]. Recently, Yedidia *et al.* [7] demonstrated an intriguing connection between BBP and certain inference methods based in statistical physics. Essentially, they demonstrated that traditional BBP algorithms can be shown to arise from approximations

of the extrema of the Bethe and Kikuchi free energies. Next, Yuille [8] derived new double-loop algorithms which are guaranteed to minimize the Bethe and Kikuchi energy functions while continuing to have close ties to the original BBP algorithms. Yuille's approach relies on a certain decomposition of the Bethe and Kikuchi free energies. In the present work, we begin with a new principle—pairwise mutual information minimization and marginal entropy maximization (MIME)—and derive a new energy function which is shown to be equivalent to the Bethe free energy. After demonstrating this connection, we derive a family of free energies closely related to the MIME principle which also shown to be equivalent, when constraint satisfaction is exact, to the Bethe free energy. For each member in this family of energy functions , we derive a new algorithm that is guaranteed to converge to a local minimum. Moreover, the resulting form of the algorithm is very simple despite the somewhat unwieldy nature of the algebraic development. Preliminary comparisons of the new algorithm with BBP were carried out on spin glass-like problems and indicate that the new algorithm is convergent when BBP is not. However, the effectiveness of the new algorithms remains to be seen.

## 2   Bethe free energy and the MIME principle

In this section, we show that the Bethe free energy can be interpreted as pairwise mutual information minimization and marginal entropy maximization.

The Bethe free energy for Bayesian belief propagation is written as

$$
\begin{aligned}
F_{\text{Bethe}}(\{p_{ij}, p_i, \gamma_{ij}, \lambda_{ij}\}) = \\
\sum_{ij:i>j} \sum_{x_i,x_j} p_{ij}(x_i,x_j) \log \frac{p_{ij}(x_i,x_j)}{\phi_{ij}(x_i,x_j)} - \sum_i (n_i - 1) \sum_{x_i} p_i(x_i) \log \frac{p_i(x_i)}{\psi_i(x_i)} \\
+ \sum_{ij:i>j} \sum_{x_j} \lambda_{ij}(x_j)[\sum_{x_i} p_{ij}(x_i,x_j) - p_j(x_j)] \\
+ \sum_{ij:i>j} \sum_{x_i} \lambda_{ji}(x_i)[\sum_{x_j} p_{ij}(x_i,x_j) - p_i(x_i)] \\
+ \sum_{ij:i>j} \gamma_{ij}(\sum_{x_i,x_j} p_{ij}(x_i,x_j) - 1)
\end{aligned}
\tag{1}
$$

where $\phi_{ij}(x_i,x_j) \stackrel{\text{def}}{=} \psi_{ij}(x_i,x_j)\psi_i(x_i)\psi_j(x_j)$ and $n_i$ is the number of neighbors of node $i$. Link functions $\psi_{ij} > 0$ are available relational data between nodes $i$ and $j$. The singleton function $\psi_i$ is also available at each node $i$. The double summation $\sum_{ij:i>j}$ is carried out only over the nodes that are connected. The Lagrange parameters $\{\lambda_{ij}, \gamma_{ij}\}$ are needed in the Bethe free energy (1) to satisfy the following constraints relating the joint probabilities $\{p_{ij}\}$ with the marginals $\{p_i\}$:

$$
\sum_{x_i} p_{ij}(x_i,x_j) = p_j(x_j), \ \ \sum_{x_j} p_{ij}(x_i,x_j) = p_i(x_i), \ \text{and} \ \sum_{x_i,x_j} p_{ij}(x_i,x_j) = 1. \tag{2}
$$

The pairwise mutual information is defined as

$$
MI_{ij} = \sum_{x_i,x_j} p_{ij}(x_i,x_j) \log \frac{p_{ij}(x_i,x_j)}{p_i(x_i)p_j(x_j)} \tag{3}
$$

The mutual information is *minimized* when the joint probability $p_{ij}(x_i,x_j) = p_i(x_i)p_j(x_j)$ or equivalently when nodes $i$ and $j$ are independent. When nodes $i$ and $j$ are connected via a non-separable link $\psi_{ij}(x_i,x_j)$ they will not be independent. We now state the MIME principle.

**Statement of the MIME principle:** Maximize the marginal entropy and minimize the pairwise mutual information using the available marginal and pairwise link function expectations while satisfying the joint probability constraints.

The pairwise MIME principle leads to the following free energy:

$$F_{\text{MIME}}(\{p_{ij}, p_i, \gamma_{ij}, \lambda_{ij}\}) =$$

$$\sum_{ij:i>j} \sum_{x_i,x_j} p_{ij}(x_i, x_j) \log \frac{p_{ij}(x_i,x_j)}{p_i(x_i)p_j(x_j)} + \sum_i \sum_{x_i} p_i(x_i) \log p_i(x_i)$$

$$- \sum_{ij:i>j} \sum_{x_i,x_j} p_{ij}(x_i, x_j) \log \psi_{ij}(x_i, x_j) - \sum_i \sum_{x_i} p_i(x_i) \log \psi_i(x_i)$$

$$+ \sum_{ij:i>j} \sum_{x_j} \lambda_{ij}(x_j)[\sum_{x_i} p_{ij}(x_i, x_j) - p_j(x_j)]$$

$$+ \sum_{ij:i>j} \sum_{x_i} \lambda_{ji}(x_i)[\sum_{x_j} p_{ij}(x_i, x_j) - p_i(x_i)]$$

$$+ \sum_{ij:i>j} \gamma_{ij}(\sum_{x_i,x_j} p_{ij}(x_i, x_j) - 1). \tag{4}$$

In the above free energy, we minimize the pairwise mutual information and maximize the marginal entropies. The singleton and pairwise link functions are additional information which do not allow the system to reach its "natural" equilibrium—a uniform i.i.d. distribution on the nodes. The Lagrange parameters enforce the constraints between the pairwise and marginal probabilities. These constraints are the same as in the Bethe free energy (1). Note that the Lagrange parameter terms vanish if the constraints in (2) are exactly satisfied. This is an important point when considering equivalences between different energy functions.

**Lemma 1** Provided the constraints in (2) are *exactly* satisfied, the MIME free energy in (4) is equivalent to the Bethe free energy in (1).

**Proof:** Using the fact that constraint satisfaction is exact and using the identity $p_{ij}(x_i, x_j) = p_{ji}(x_j, x_i)$, we may write

$$- \sum_{ij:i>j} \sum_{x_i,x_j} p_{ij}(x_i, x_j) \log p_i(x_i)p_j(x_j) = - \sum_{ij:i\neq j} \sum_{x_i,x_j} p_{ij}(x_i, x_j) \log p_i(x_i)$$

$$= - \sum_i n_i \sum_{x_i} p_i(x_i) \log p_i(x_i),$$

$$\text{and } \sum_{ij:i>j} \sum_{x_i,x_j} p_{ij}(x_i, x_j) \log \psi_i(x_i)\psi_j(x_j) = \sum_i n_i \sum_{x_i} p_i(x_i) \log \psi_i(x_i). \tag{5}$$

We have shown that a marginal entropy term emerges from the mutual information term in (4) when constraint satisfaction is exact. Collecting the marginal entropy terms together and rearranging the MIME free energy in (4), we get the Bethe free energy in (1).

## 3 A family of decompositions of the Bethe free energy

Recall that the Bethe free energy and the energy function resulting from application of the MIME principle were shown to be equivalent. However, the MIME energy function is merely one particular decomposition of the Bethe free energy. As Yuille mentions [8], many decompositions are possible. The main motivation for considering alternative decompositions is for algorithmic reasons. *We believe that certain decompositions may be more effective than others. This belief is based on our previous experience with closely related deterministic annealing algorithms* [3, 2]. In this section, we derive a family of free energies that are equivalent to the Bethe free energy provided constraint satisfaction is exact. The family of free energies is inspired by and closely related to the MIME free energy in (4).

**Lemma 2** The following family of energy functions indexed by the free parameters $\delta > 0$ and $\{\xi_i\}$ is equivalent to the original Bethe free energy (1) provided the

constraints in (2) are exactly satisfied and the parameters $q$ and $r$ are set to $\{q_i = (1-\delta)n_i\}$ and $\{r_i = 1 - n_i\xi_i\}$ respectively.

$$F_{\text{equiv}}(\{p_{ij}, p_i, \gamma_{ij}, \lambda_{ij}\}) =$$
$$\sum_{ij:i>j} \sum_{x_i,x_j} p_{ij}(x_i,x_j) \log \frac{p_{ij}(x_i,x_j)}{[\sum_{x_j} p_{ij}(x_i,x_j)]^\delta [\sum_{x_i} p_{ij}(x_i,x_j)]^\delta}$$
$$+ \sum_i \sum_{x_i} p_i(x_i) \log p_i(x_i) - \sum_i q_i \sum_{x_i} p_i(x_i) \log p_i(x_i)$$
$$- \sum_{ij:i>j} \sum_{x_i,x_j} p_{ij}(x_i,x_j) \log \psi_{ij}(x_i,x_j) \psi_i^{\xi_i}(x_i) \psi_j^{\xi_j}(x_j)$$
$$- \sum_i r_i \sum_{x_i} p_i(x_i) \log \psi_i(x_i)$$
$$+ \sum_{ij:i>j} \sum_{x_j} \lambda_{ij}(x_j)[\sum_{x_i} p_{ij}(x_i,x_j) - p_j(x_j)]$$
$$+ \sum_{ij:i>j} \sum_{x_i} \lambda_{ji}(x_i)[\sum_{x_j} p_{ij}(x_i,x_j) - p_i(x_i)]$$
$$+ \sum_{ij:i>j} \gamma_{ij}(\sum_{x_i,x_j} p_{ij}(x_i,x_j) - 1). \tag{6}$$

In (6), the first term is no longer the pairwise mutual information as in (4). And unlike (4), $p_i(x_i)$ no longer appears in the pairwise mutual information-like term.

**Proof:** We selectively substitute $\sum_{x_i} p_{ij}(x_i,x_j) = p_j(x_j)$ and $\sum_{x_j} p_{ij}(x_i,x_j) = p_i(x_i)$ to show the equivalence. First

$$\sum_{ij:i>j} \sum_{x_i,x_j} p_{ij}(x_i,x_j) \log[\sum_{x_j} p_{ij}(x_i,x_j)]^\delta [\sum_{x_i} p_{ij}(x_i,x_j)]^\delta = \delta \sum_i n_i \sum_{x_j} p_i(x_i) \log p_i(x_i),$$
$$\sum_{ij:i>j} \sum_{x_i,x_j} p_{ij}(x_i,x_j) \log \psi_i^{\xi_i}(x_i) \psi_j^{\xi_j}(x_j) = \sum_i n_i \xi_i \sum_{x_j} p_i(x_i) \log \psi_i(x_i). \tag{7}$$

Substituting the identities in (7) into (6), we see that the free energies are algebraically equivalent.

# 4    A family of algorithms for belief propagation

We now derive descent algorithms for the family of energy functions in (6). All the algorithms are guaranteed to converge to a local minimum of (6) under mild assumptions regarding the number of fixed points. For each member in the family of energy functions, there is a corresponding descent algorithm. Since the form of the free energy in (6) is complex and precludes easy minimization, we use algebraic (Legendre) transformations [1] to simplify the optimization.

$$-\sum_{x_j} p_{ij}(x_i,x_j) \log \sum_{x_j} p_{ij}(x_i,x_j) =$$
$$\min_{\sigma_{ji}(x_i)} - \sum_{x_j} p_{ij}(x_i,x_j) \log \sigma_{ji}(x_i) + \sigma_{ji}(x_i) - \sum_{x_j} p_{ij}(x_i,x_j)$$

$$-\sum_{x_i} p_{ij}(x_i,x_j) \log \sum_{x_i} p_{ij}(x_i,x_j) =$$
$$\min_{\sigma_{ij}(x_j)} - \sum_{x_i} p_{ij}(x_i,x_j) \log \sigma_{ij}(x_j) + \sigma_{ij}(x_j) - \sum_{x_i} p_{ij}(x_i,x_j)$$
$$-p_i(x_i) \log p_i(x_i) = \min_{\rho_i(x_i)} -p_i(x_i) \log \rho_i(x_i) + \rho_i(x_i) - p_i(x_i). \tag{8}$$

We now apply the above algebraic transforms. The new free energy is (after some algebraic manipulations)

$$F_{\text{equiv}}(\{p_{ij}, p_i, \sigma_{ij}, \rho_i, \gamma_{ij}, \lambda_{ij}\}) = \sum_{ij:i>j} \sum_{x_i,x_j} p_{ij}(x_i,x_j) \log \frac{p_{ij}(x_i,x_j)}{\sigma_{ji}^\delta(x_i)\sigma_{ij}^\delta(x_j)}$$

$$+\delta \sum_{ij:i\neq j} \sum_{x_i} \sigma_{ij}(x_j) + \sum_i \sum_{x_i} p_i(x_i) \log \frac{p_i(x_i)}{\rho_i^{q_i}(x_i)} + \sum_i q_i \sum_{x_i} \rho_i(x_i)$$

$$- \sum_{ij:i>j} \sum_{x_i,x_j} p_{ij}(x_i,x_j) \log \psi_{ij}(x_i,x_j)\psi_i^{\xi_i}(x_i)\psi_j^{\xi_j}(x_j) - \sum_i r_i \sum_{x_i} p_i(x_i) \log \psi_i(x_i)$$

$$+ \sum_{ij:i>j} \sum_{x_j} \lambda_{ij}(x_j)[\sum_{x_i} p_{ij}(x_i,x_j) - p_j(x_j)] + \sum_{ij:i>j} \sum_{x_i} \lambda_{ji}(x_i)[\sum_{x_j} p_{ij}(x_i,x_j) - p_i(x_i)]$$

$$+ \sum_{ij:i>j} \gamma_{ij}(\sum_{x_i,x_j} p_{ij}(x_i,x_j) - 1). \qquad (9)$$

We continue to keep the parameters $\{q_i\}$ and $\{r_i\}$ in (9). However, from Lemma 2, we know that the equivalence of (9) to the Bethe free energy is predicated upon appropriate setting of these parameters. In the rest of the paper, we continue to use $q$ and $r$ for the sake of notational simplicity.

Despite the introduction of new variables via Legendre transforms, the optimization problem in (9) is still a minimization problem over *all* the variables. The algebraically transformed energy function in (9) is separately convex w.r.t. $\{p_{ij}, p_i\}$ and w.r.t. $\{\sigma_{ij}, \rho_i\}$ provided $\delta \in [0,1]$. Since the overall energy function is not convex w.r.t. all the variables, we pursue an alternating algorithm strategy similar to the double loop algorithm in Yuille [8]. The basic idea is to separately minimize w.r.t. the variables $\{\sigma_{ij}, \rho_i\}$ and the variables $\{p_{ij}, p_i\}$. The linear constraints in (2) are enforced when minimizing w.r.t the latter *and do not affect the convergence properties of the algorithm since the energy function w.r.t. $\{p_{ij}, p_i\}$ is convex*.

We evaluate the fixpoints of $\{\sigma_{ij}, \rho_i\}$. Note that (9) is convex w.r.t. $\{\sigma_{ij}, \rho_i\}$.

$$\sigma_{ij}(x_j) = \sum_{x_i} p_{ij}(x_i,x_j), \ \ \sigma_{ji}(x_i) = \sum_{x_j} p_{ij}(x_i,x_j), \ \text{and } \rho_i(x_i) = p_i(x_i). \qquad (10)$$

The fixpoints of $\{p_{ij}, p_i\}$ are evaluated next. Note that (9) is convex w.r.t. $\{p_{ij}, p_i\}$.

$$p_{ij}(x_i,x_j) = \sigma_{ji}^{\delta}(x_i)\sigma_{ij}^{\delta}(x_j)\psi_{ij}(x_i,x_j)\psi_i^{\xi_i}(x_i)\psi_j^{\xi_j}(x_j)e^{-\lambda_{ij}(x_j)-\lambda_{ji}(x_i)-\gamma_{ij}-1}$$

$$p_i(x_i) = \rho_i^{q_i}(x_i)\psi_i^{r_i}(x_i)e^{\sum_k \lambda_{ki}(x_i)-1}. \qquad (11)$$

The constraint satisfaction equations from (2) can be rewritten as

$$\sum_{x_j} p_{ij}(x_i,x_j) = p_i(x_i) \Rightarrow$$

$$e^{2\lambda_{ji}(x_i)} = \frac{\sum_{x_j} \sigma_{ji}^{\delta}(x_i)\sigma_{ij}^{\delta}(x_j)\psi_{ij}(x_i,x_j)\psi_i^{\xi_i}(x_i)\psi_j^{\xi_j}(x_j)e^{-\lambda_{ij}(x_j)-\gamma_{ij}-1}}{\rho_i^{q_i}(x_i)\psi_i^{r_i}(x_i)e^{\sum_{k\neq j}\lambda_{ki}(x_i)-1}} \qquad (12)$$

Similar relations can be obtained for the other constraints in (2). Consider a Lagrange parameter update sequence where the Lagrange parameter currently being updated is tagged as "new" with the rest designated as "old." We can then rewrite the Lagrange parameter updates using "old" and "new" values. Please note that each Lagrange parameter update corresponds to one of the constraints in (2). It can be shown that the iterative update of the Lagrange parameters is guaranteed to converge to the unique solution of (2) [8]. While rewriting (12), we multiply the left and right sides with $e^{-2\lambda_{ji}^{\text{old}}(x_i)}$.

$$e^{2\lambda_{ji}^{\text{new}}(x_i)-2\lambda_{ji}^{\text{old}}(x_i)} =$$

$$\frac{\sum_{x_j} \sigma_{ji}^{\delta}(x_i)\sigma_{ij}^{\delta}(x_j)\psi_{ij}(x_i,x_j)\psi_i^{\xi_i}(x_i)\psi_j^{\xi_j}(x_j)e^{-\lambda_{ij}^{\text{old}}(x_j)-\lambda_{ji}^{\text{old}}(x_i)-\gamma_{ij}^{\text{old}}-1}}{\rho_i^{q_i}(x_i)\psi_i^{r_i}(x_i)e^{\sum_k \lambda_{ki}^{\text{old}}(x_i)-1}}. \qquad (13)$$

Using (11), we relate each Lagrange parameter update with an update of $p_{ij}(x_i, x_j)$ and $p_i(x_i)$. We again invoke the "old" and "new" designations, this time on the probabilities. From (11), (12) and (13), we write the joint probability update

$$\frac{p_{ij}^{\text{new}}(x_i, x_j)}{p_{ij}^{\text{old}}(x_i, x_j)} = e^{-\lambda_{ji}^{\text{new}}(x_i) + \lambda_{ji}^{\text{old}}(x_i)} = \sqrt{\frac{p_i^{\text{old}}(x_i)}{\sum_{x_j} p_{ij}^{\text{old}}(x_i, x_j)}} \tag{14}$$

and for the marginal probability update

$$\frac{p_i^{\text{new}}(x_i)}{p_i^{\text{old}}(x_i)} = e^{\lambda_{ji}^{\text{new}}(x_i) - \lambda_{ji}^{\text{old}}(x_i)} = \sqrt{\frac{\sum_{x_j} p_{ij}^{\text{old}}(x_i, x_j)}{p_i^{\text{old}}(x_i)}}. \tag{15}$$

From (14) and (15), the update equations for the probabilities are

$$p_{ij}^{\text{new}}(x_i, x_j) = p_{ij}^{\text{old}}(x_i, x_j)\sqrt{\frac{p_i^{\text{old}}(x_i)}{\sum_{x_j} p_{ij}^{\text{old}}(x_i, x_j)}}, p_i^{\text{new}}(x_i) = \sqrt{p_i^{\text{old}}(x_i) \sum_{x_j} p_{ij}^{\text{old}}(x_i, x_j)} \tag{16}$$

With the probability updates in place, we may write down new algorithms minimizing the family of Bethe equivalent free energies using only probability updates. The update equations (16) can be seen to satisfy the first constraint in (2). Similar update equations can be derived for the other constraints in (2). For each Lagrange parameter update, an equivalent, simultaneous probability (joint and marginal) update can be derived similar to (16). The overall family of algorithms can be summarized as shown in the pseudocode. Despite the unwieldy algebraic development preceding it, the algorithm is very simple and straightforward.

Set free parameters $\delta \in [0, 1]$ and $\{\xi_i\}$.
Initialize $\{p_{ij}, p_i\}$. Set $\{q_i = (1 - \delta)n_i\}$ and $\{r_i = 1 - n_i\xi_i\}$.
**Begin A: Outer Loop**
$\quad \sigma_{ij}(x_j) \leftarrow \sum_{x_i} p_{ij}(x_i, x_j)$
$\quad \sigma_{ji}(x_i) \leftarrow \sum_{x_j} p_{ij}(x_i, x_j)$
$\quad \rho_i(x_i) \leftarrow p_i(x_i)$
$\quad p_{ij}(x_i, x_j) \leftarrow \sigma_{ji}^{\delta}(x_i)\sigma_{ij}^{\delta}(x_j)\psi_{ij}(x_i, x_j)\psi_i^{\xi_i}(x_i)\psi_j^{\xi_j}(x_j)$
$\quad p_i(x_i) \leftarrow \rho_i^{q_i}(x_i)\psi_i^{r_i}(x_i)$
$\quad$ **Begin B: Inner Loop:** Do B until $\frac{1}{N}\sum_{ij:i>j}[(\sum_{x_j} p_{ij}(x_i, x_j) - p_i(x_i))^2 + (\sum_{x_i} p_{ij}(x_i, x_j) - p_j(x_j))^2] < c_{\text{thr}}$
$\quad\quad$ Simultaneously update $p_{ij}(x_i, x_j)$ and $p_i(x_i)$ below.
$\quad\quad p_{ij}(x_i, x_j) \leftarrow p_{ij}(x_i, x_j)\sqrt{\frac{p_i(x_i)}{\sum_{x_j} p_{ij}(x_i, x_j)}}$
$\quad\quad p_i(x_i) \leftarrow \sqrt{p_i(x_i) \sum_{x_j} p_{ij}(x_i, x_j)}$
$\quad\quad$ Simultaneously update $p_{ij}(x_i, x_j)$ and $p_j(x_j)$ below.
$\quad\quad p_{ij}(x_i, x_j) \leftarrow p_{ij}(x_i, x_j)\sqrt{\frac{p_j(x_j)}{\sum_{x_i} p_{ij}(x_i, x_j)}}$
$\quad\quad p_j(x_j) \leftarrow \sqrt{p_j(x_j) \sum_{x_i} p_{ij}(x_i, x_j)}$
$\quad\quad$ Normalize $p_{ij}(x_i, x_j)$.
$\quad\quad p_{ij}(x_i, x_j) \leftarrow \frac{p_{ij}(x_i, x_j)}{\sum_{x_i, x_j} p_{ij}(x_i, x_j)}$

**End B**

**End A**

In the above family of algorithms, the MIME algorithm corresponds to free parameter settings $\delta = 1$ and $\xi_i = 0$ which in turn lead to parameter settings $q_i = 0$ and $r_i = 1$. The Yuille [8] double loop algorithm corresponds to the free parameter settings $\delta = 0$ and $\xi_i = 0$ which in turn leads to parameter settings $q_i = n_i$ and $r_i = 1$. *A crucial point is that the energy function for every valid parameter setting is equivalent to the Bethe free energy provided constraint satisfaction is exact.* The inner loop constraint satisfaction threshold parameter $c_{\text{thr}}$ setting is very important in this regard. We are obviously not restricted to the MIME parameter settings. At this early stage of exploration of the inter-relationships between Bayesian belief propagation and inference methods based in statistical physics [7], it is premature to speculate regarding the "best" parameter settings for $\delta$ and $\{\xi_i\}$. Most likely, the effectiveness of the algorithms will vary depending on the problem setting which enters into the formulation via the link functions $\{\psi_{ij}\}$ and the singleton functions $\{\psi_i\}$.

## 5 Results

We implemented the family of algorithms in C++ and conducted tests on locally connected 50 node graphs and binary state variables. The $\psi_i(x_i)$ and $\psi_{ij}(x_i, x_j)$ are of the form $e^{\pm h_i}$ and $e^{\pm h_{ij}}$ where $h_i$ and $h_{ij}$ are drawn from uniform distributions (in the interval $[-1, 1]$). Provided the constraint satisfaction theshold parameter $c_{\text{thr}}$ was set low enough, the algorithm (for $\delta = 1$ and other parameter settings as described in Figure 1) exhibited monotonic convergence. Figure 2 shows the number of inner loop iterations corresponding to different settings of the constraint satisfaction threshold parameter. We also implemented the BBP algorithm and empirically observed that it often did not converge for these graphs. These results are quite preliminary and far more validation experiments are required. However, they provide a proof of concept for our approach.

## 6 Conclusion

We began with the MIME principle and showed the equivalence of the MIME-based free energy to the Bethe free energy assuming constraint satisfaction to be exact. Then, we derived new decompositions of the Bethe free energy inspired by the MIME principle, and driven by our belief that certain decompositions may be more effective than others. We then derived a convergent algorithm for each member in the family of MIME-based decompositions. It remains to be seen if the MIME-based algorithms are efficient for a reasonable class of problems. While the MIME-based algorithms derived here use closed-form solutions in the constraint satisfaction inner loop, it may turn out that the inner loop is better handled using preconditioned gradient-based descent algorithms. And it is important to explore the inter-relationships between the convergent MIME-based descent algorithms and other recent related approaches with interesting convergence properties [4, 5].

## References

[1] E. Mjolsness and C. Garrett. Algebraic transformations of objective functions. *Neural Networks*, 3:651–669, 1990.

[2] A. Rangarajan. Self annealing and self annihilation: unifying deterministic annealing and relaxation labeling. *Pattern Recognition*, 33:635–649, 2000.

[3] A. Rangarajan, S. Gold, and E. Mjolsness. A novel optimizing network architecture with applications. *Neural Computation*, 8(5):1041–1060, 1996.

[4] Y. W. Teh and M. Welling. Passing and bouncing messages for generalized inference. Technical Report GCNU 2001-01, Gatsby Computational Neuroscience Unit, University College, London, 2001.

[5] M. Wainwright, T. Jaakola, and A. Willsky. Tree-based reparameterization framework for approximate estimation of stochastic processes on graphs with cycles. Technical Report LIDS P-2510, MIT, Cambridge, MA, 2001.

[6] Y. Weiss. Correctness of local probability propagation in graphical models with loops. *Neural Computation*, 12:1–41, 2000.

[7] J. S. Yedidia, W. T. Freeman, and Y. Weiss. Bethe free energy, Kikuchi approximations and belief propagation algorithms. In *Advances in Neural Information Processing Systems 13*, Cambridge, MA, 2001. MIT Press.

[8] A. L. Yuille. A double loop algorithm to minimize the Bethe and Kikuchi free energies. *Neural Computation*, 2001. (submitted).

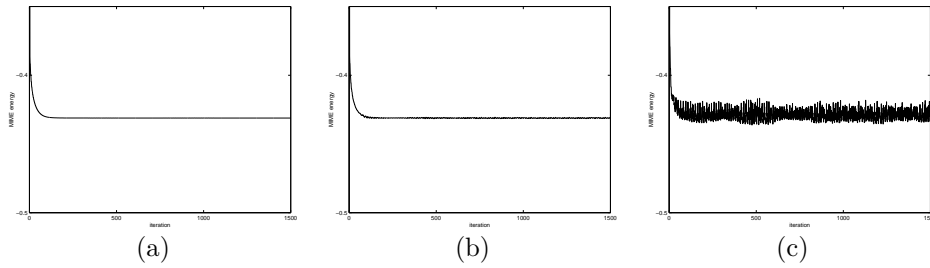

Figure 1: **MIME energy versus outer loop iteration:** 50 node, local topology, $\delta = 1$. Constraint satisfaction threshold parameter $c_{\text{thr}}$ was set to (a) $10^{-8}$ (b) $10^{-4}$ (c) $10^{-2}$

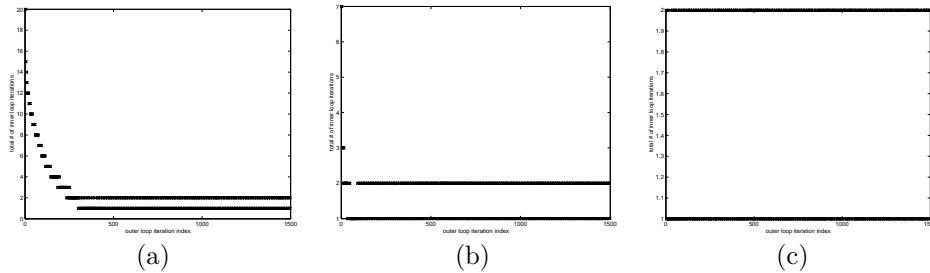

Figure 2: **Inner loop iterations versus outer loop:** 50 node, local topology, $\delta = 1$. Constraint satisfaction threshold parameter $c_{\text{thr}}$ was set to (a) $10^{-8}$ (b) $10^{-4}$ (c) $10^{-2}$